# Understanding stepwise generalization of Support Vector Machines: a toy model

**Sebastian Risau-Gusman and Mirta B. Gordon**
DRFMC/SPSMS CEA Grenoble, 17 av. des Martyrs
38054 Grenoble cedex 09, France

## Abstract

In this article we study the effects of introducing structure in the input distribution of the data to be learnt by a simple perceptron. We determine the learning curves within the framework of Statistical Mechanics. Stepwise generalization occurs as a function of the number of examples when the distribution of patterns is highly anisotropic. Although extremely simple, the model seems to capture the relevant features of a class of Support Vector Machines which was recently shown to present this behavior.

## 1 Introduction

A new approach to learning has recently been proposed as an alternative to feedforward neural networks: the Support Vector Machines (SVM) [1]. Instead of trying to learn a non linear mapping between the input patterns and internal representations, like in multilayered perceptrons, the SVMs choose *a priori* a non-linear kernel that transforms the input space into a high dimensional feature space. In binary classification tasks like those considered in the present paper, the SVMs look for linear separation with optimal margin in feature space. The main advantage of SVMs is that learning becomes a convex optimization problem. The difficulties of having many local minima that hinder the process of training multilayered neural networks is thus avoided. One of the questions raised by this approach is why SVMs do not overfit the data in spite of the extremely large dimensions of the feature spaces considered.

Two recent theoretical papers [2, 3] studied a family of SVMs with the tools of Statistical Mechanics, predicting typical properties in the limit of large dimensional spaces. Both papers considered mappings generated by polynomial kernels, and more specifically quadratic ones. In these, the input vectors $\mathbf{x} \in \mathbf{R}^N$ are transformed to $N(N + 1)/2$-dimensional feature vectors $\Phi(\mathbf{x})$. More precisely, the mapping $\Phi_1(\mathbf{x}) = (\mathbf{x}, x_1\mathbf{x}, x_2\mathbf{x}, \cdots, x_k\mathbf{x})$ has been studied in [3] as a function of $k$, the number of quadratic features, and $\Phi_2(\mathbf{x}) = (\mathbf{x}, x_1\mathbf{x}/N, x_2\mathbf{x}/N, \cdots, x_N\mathbf{x}/N)$ has been considered in [2], leading to different results. These mappings are particular cases of quadratic kernels. In particular, in the case of learning quadratically separable tasks with mapping $\Phi_2$, the generalization error decreases up to a lower bound for a number of examples proportional to $N$, followed by a further decrease if the number of examples increases proportionally to the dimension of the feature

space, *i.e.* to $N^2$. In fact, this behavior is not specific of the SVMs. It also arises in the typical case of Gibbs learning (defined below) in quadratic feature spaces [4]: on increasing the training set size, the quadratic components of the discriminating surface are learnt after the linear ones. In the case of learning linearly separable tasks in quadratic feature spaces, the effect of overfitting is harmless, as it only slows down the decrease of the generalization error with the training set size. In the case of mapping $\Phi_1$, overfitting is dramatic, as the generalization error at any given training set size increases with the number $k$ of features.

The aim of the present paper is to understand the influence of the mapping scaling-factor on the generalization performance of the SVMs. To this end, it is worth to remark that features $\Phi_2$ may be obtained by compressing the quadratic subspace of $\Phi_1$ by a fixed factor. In order to mimic this contraction, we consider a linearly separable task in which the input patterns have a highly anisotropic distribution, so that the variance in one subspace is much smaller than in the orthogonal directions. We show that in this simple toy model, the generalization error as a function of the training set size exhibits a cross-over between two different behaviors: a rapid decrease corresponding to learning the components in the uncompressed space, followed by a slow improvement in which mainly the components in the compressed space are learnt. The latter would correspond, in this highly stylized model, to learning the scaled quadratic features in the SVM with mapping $\Phi_2$.

The paper is organized as follows: after a short presentation of the model, we describe the main steps of the Statistical Mechanics calculation. The order parameters caracterizing the properties of the learning process are defined, and their evolution as a function of the training set size is analyzed. The two regimes of the generalization error are described, and we determine the training set size per input dimension at the crossover, as a function of the pertinent parameters. Finally we discuss our results, and their relevance to the understanding of the generalization properties of SVMs.

## 2   The model

We consider the problem of learning a binary classification task from examples. The training data set $\mathcal{D}_\alpha$ contains $P = \alpha N$ $N$-dimensional patterns $(\xi^\mu, \tau^\mu)$ $(\mu = 1, \cdots, P)$ where $\tau^\mu = \text{sign}(\xi^\mu \cdot \mathbf{w}^*)$ is given by a teacher of weights $\mathbf{w}^* = (w_1, w_2, ...., w_n)$. Without any loss of generality we consider normalized teachers: $\mathbf{w}^* \cdot \mathbf{w}^* = N$. We assume that the components $\xi_i$, $(i = 1, \cdots, N)$ of the input patterns $\xi$ are independent, identically distributed random variables drawn from a zero-mean gaussian distribution, with variance $\sigma$ along $N_c$ directions and unit variance in the $N_u$ remaining ones $(N_c + N_u = N)$:

$$P(\xi) = \prod_{i \in N_c} \frac{1}{\sqrt{2\pi\sigma^2}} \exp\left(-\frac{\xi_i^2}{2\sigma^2}\right) \prod_{i \in N_u} \frac{1}{\sqrt{2\pi}} \exp\left(-\frac{\xi_i^2}{2}\right). \qquad (1)$$

We take $\sigma < 1$ without any loss of generality, as the case $\sigma > 1$ may be deduced from the former through a straightforward rescaling of $N_c$ and $N_u$. Hereafter, the subspace of dimension $N_c$ and variance $\sigma$ will be called *compressed* subspace. The corresponding orthogonal subspace, of dimension $N_u = N - N_c$, will be called *uncompressed* subspace.

We study the typical generalization error of a student perceptron learning the classification task, using the tools of Statistical Mechanics. The pertinent cost function

is the number of misclassified patterns:

$$E(\mathbf{w}; \mathcal{D}_\alpha) = \sum_{\mu=1}^{P} \Theta(-\tau^\mu \, \xi^\mu \cdot \mathbf{w}),$$ (2)

The weight vectors in version space correspond to a vanishing cost (2). Choosing a **w** at random from the *a posteriori* distribution

$$P(\mathbf{w}|\mathcal{D}_\alpha) = Z^{-1} P_0(\mathbf{w}) \exp\left(-\beta E(\mathbf{w}; \mathcal{D}_\alpha)\right),$$ (3)

in the limit of $\beta \to \infty$ is called Gibbs' learning. In eq. (3), $\beta$ is equivalent to an inverse temperature in the Statistical Mechanics formulation, the cost (2) being the energy function. We assume that $P_0$, the *a priori* distribution of the weights, is uniform on the hypersphere of radius $\sqrt{N}$:

$$P_0(\mathbf{w}) = (2\pi e)^{-N/2} \, \delta(\mathbf{w} \cdot \mathbf{w} - N).$$ (4)

The normalization constant $(2\pi e)^{N/2}$ is the leading order term of the hypersphere's surface in $N$-dimensional space. $Z$ is the partition function ensuring the correct normalization of $P(\mathbf{w}|\mathcal{D}_\alpha)$:

$$Z(\beta; \mathcal{D}_\alpha) = \int d\mathbf{w} \, P_0(\mathbf{w}) \exp\left(-\beta E(\mathbf{w}; \mathcal{D}_\alpha)\right).$$ (5)

In general, the properties of the student are related to those of the free energy $F(\beta; \mathcal{D}_\alpha) = -\ln Z(\beta; \mathcal{D}_\alpha)/\beta$. In the limit $N \to \infty$ with the training set size per input dimension $\alpha \equiv P/N$ constant, the properties of the student weights become independant of the particular training set $\mathcal{D}_\alpha$. They are deduced from the averaged free energy per degree of freedom, calculated using the replica trick:

$$f(\beta) = -\frac{1}{N\beta}\overline{\ln Z(\beta; \mathcal{D}_\alpha)} = -\frac{1}{N\beta} \lim_{n \to 0} \frac{\ln \overline{Z^n(\beta; \mathcal{D}_\alpha)}}{n}$$ (6)

where the overline represents the average over $\mathcal{D}_\alpha$, composed of patterns selected according to (1). In the case of Gibbs learning, the typical behavior of any intensive quantity is obtained in the zero temperature limit $\beta \to \infty$. In this limit, only error-free solutions, with vanishing cost, have non-vanishing posterior probability (3). Thus, Gibbs learning corresponds to picking at random a student in version space, *i.e.* a vector **w** that classifies correctly the training set $\mathcal{D}_\alpha$, with a probability proportional to $P_0(\mathbf{w})$.

In the case of an isotropic pattern distribution, which corresponds to $\sigma = 1$ in (1), the properties of cost function (2) have been extensively studied [5]. The case of patterns drawn from two gaussian clusters in which the symmetry axis of the clusters is the same [6] and different [7] from the teacher's axis, have recently been addressed. Here we consider the problem where, instead of having a single direction along which the patterns' distribution is contracted (or expanded), there is a *finite* fraction of compressed dimensions. In this case, all the properties of the student's perceptron may be expressed in terms of the following order parameters, that have to satisfy corresponding extremum conditions of the free energy:

$$\tilde{q}_c^{ab} = \langle \frac{1}{N} \sum_{i \in N_c} w_{ia} w_{ib} \rangle$$ (7)

$$\tilde{q}_u^{ab} = \langle \frac{1}{N} \sum_{i \in N_u} w_{ia} w_{ib} \rangle$$ (8)

$$\tilde{R}_c^a = \langle \frac{1}{N} \sum_{i \in N_c} w_{ia} w_i^* \rangle \tag{9}$$

$$\tilde{R}_u^a = \langle \frac{1}{N} \sum_{i \in N_u} w_{ia} w_i^* \rangle \tag{10}$$

$$Q^a = \langle \frac{1}{N} \sum_{i \in N_c} (w_{ia})^2 \rangle \tag{11}$$

where $\langle \cdots \rangle$ indicates the average over the posterior (3); $a, b$ are replica indices, and the subcripts $c$ and $u$ stand for compressed and uncompressed respectively. Notice that we do not impose that $Q^a$, the typical squared norm of the student's components in the compressed subspace, be equal to the corresponding teacher's norm $Q^* = \sum_{i \in N_c} (w_i^*)^2 / N$.

## 3  Order parameters and learning curves

Assuming that the order parameters are invariant under permutation of replicas, we can drop the replica indices in equations (7) to (11). We expect that this hypothesis of replica symmetry is consistent, like it is in other cases of perceptrons learning realizable tasks. The problem is thus reduced to the determination of five order parameters. Their meaning becomes clearer if we consider the following combinations:

$$q_c = \frac{\tilde{q}_c}{Q}, \tag{12}$$

$$q_u = \frac{\tilde{q}_u}{1 - Q}, \tag{13}$$

$$R_c = \frac{\tilde{R}_c}{\sqrt{Q}\sqrt{Q^*}}, \tag{14}$$

$$R_u = \frac{\tilde{R}_u}{\sqrt{1-Q}\sqrt{1-Q^*}}, \tag{15}$$

$$Q = \langle \frac{1}{N} \sum_{i \in N_c} (w_i)^2 \rangle. \tag{16}$$

$q_c$ and $q_u$ are the typical overlaps between the components of two student vectors in the compressed and the uncompressed subspaces respectively. Similarly, $R_c$ and $R_u$ are the corresponding overlaps between a typical student and the teacher. In terms of this set of parameters, the typical generalization error is $\epsilon_g = (1/\pi) \arccos R$ with

$$R = \frac{\sigma^2 R_c \sqrt{QQ^*} + R_u \sqrt{(1-Q)(1-Q^*)}}{\sqrt{\sigma^2 Q + (1-Q)}\sqrt{\sigma^2 Q^* + (1-Q^*)}}. \tag{17}$$

Given $\alpha$, the general solution to the extremum conditions depends on the three parameters of the problem, namely $\sigma$, $Q^*$ and $n_c \equiv N_c/N$. An interesting case is the one where the teacher's anisotropy is consistent with that of the pattern's distribution, i.e. $Q^* = n_c$. In this case, it easy to show that $Q = Q^*$, $q_c = R_c$ and $q_u = R_u$. Thus,

$$R = \frac{n_u R_u + \sigma^2 n_c R_c}{n_u + \sigma^2 n_c}, \tag{18}$$

where $n_u \equiv N_u/N$, $R_u$ and $R_c$ are given by the following equations:

$$\frac{R_c}{1 - R_c} = \frac{\sigma^2}{\sigma^2 n_c + n_u} \frac{\alpha}{\pi \sqrt{1 - R}} \int \mathcal{D}t \frac{\exp\left(-Rt^2/2\right)}{H(t\sqrt{R})}, \tag{19}$$

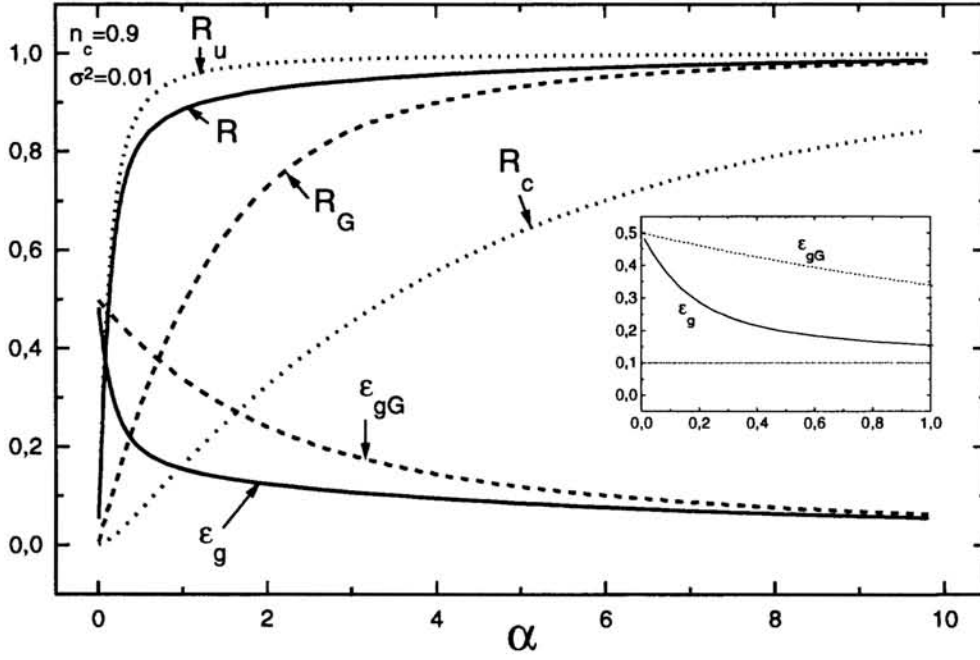

Figure 1: Order parameters and generalization error for the case $Q^* = n_c = 0.9$, $\sigma^2 = 10^{-2}$. The curves for the case of spherically distributed patterns is shown for comparison. The inset shows the first step of learning and its *plateau* (see text).

$$\frac{R_c}{1 - R_c} = \sigma^2 \frac{R_u}{1 - R_u}. \tag{20}$$

where $\mathcal{D}t = dt\, e^{-t^2/2}/\sqrt{2\pi}$ and $H(x) = \int_x^\infty \mathcal{D}t$. If $\sigma^2 = 1$, we recover the equations corresponding to Gibbs learning of isotropic pattern distributions [5].

The order parameters are represented as a function of $\alpha$ on figure 1, for a particular choice of $n_c$ and $\sigma$. $R_u$ grows much faster than $R_c$, meaning that it is easier to learn the components of the uncompressed space. As a result, $R$ (and therefore the generalization error $\epsilon_g$) presents a cross-over between two behaviors. At small $\alpha$, both $R_u \ll 1$ and $R_c \ll 1$, so that $R(\alpha, \sigma^2) = R_G(\alpha(n_u + \sigma^4 n_c)/(n_u + \sigma^2 n_c)^2)$ where $R_G$ is the overlap for Gibbs learning with an isotropic ($\sigma^2 = 1$) distribution [5]. Learning the anisotropic distribution is faster (in $\alpha$) than learning the isotropic one. If $\sigma \ll 1$ the anisotropy is very large and $R$ increases like $R_G$ but with an effective training set size per input dimension $\sim \alpha/n_u > \alpha$. On increasing $\alpha$, there is an intermediate regime in which $R_u$ increases but $R_c \ll 1$, so that $R \simeq R_u n_u/(n_u + \sigma^2 n_c)$. The corresponding generalization error seems to reach a *plateau* corresponding to $R_u = 1$ and $R_c = 0$. At $\alpha \gg 1$, $R(\alpha, \sigma^2) \simeq R_G(\alpha)$, the asymptotic behavior is independent of the details of the distribution, like in [7]. The crossover between these two regimes, when $\sigma^2 \ll 1$, occurs at $\alpha_0 \approx \sqrt{2(n_u + \sigma^2 n_c)/(\sigma^2 n_c)}$.

The cases $Q^* = 1$ and $Q^* = 0$ are also of interest. $Q^* = 1$ corresponds to a teacher having all the weights components in the compressed subspace, whereas $Q^* = 0$

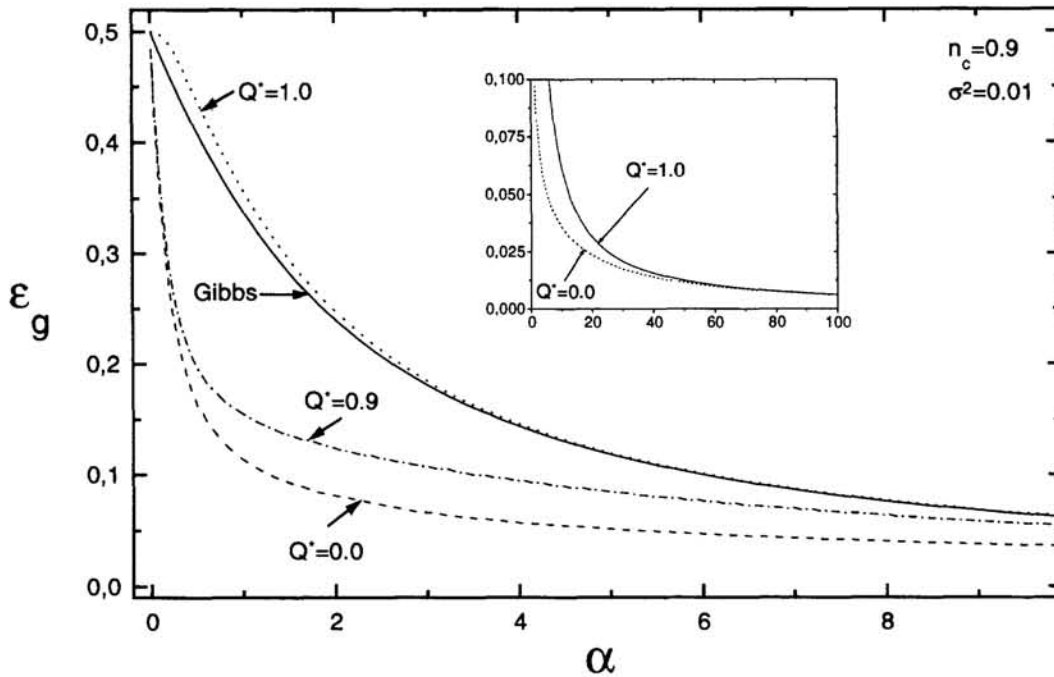

Figure 2: Generalization errors as a function of $\alpha$ for different teachers ($Q^* = 1$, $Q^* = 0.9$ and $Q^* = 1$), for the case $n_c = 0.9$ and $\sigma^2 = 10^{-2}$. The curve for spherically distributed patterns [5] is included for comparison. The inset shows the large alpha behaviors.

corresponds to a teacher orthogonal to the compressed subspace, *i.e.* with all the components in the uncompressed subspace. They correspond respectively to tasks where either the uncompressed or the compressed components are irrelevant for the patterns' classification. In Figure 2 we show all the generalization error curves, including the generalization error $\epsilon_{gG}$ for a uniform distribution [5] for comparison. The behaviour of $\epsilon_g(\alpha)$ is very sensitive to the value of $Q^*$. If $Q^* = 1$, the teacher is in the compressed subspace where learning is difficult. Consequently, $\epsilon_g(\alpha) > \epsilon_{gG}(\alpha)$ as expected. On the contrary, for $Q^* = 0$, only the components in the uncompressed space are relevant for the classification task. In this subspace learning is easy and $\epsilon_g(\alpha) < \epsilon_{gG}(\alpha)$. At $Q^* \neq 0, 1$ there is a crossover between these regimes, as already discussed. All the curves merge in the asymptotic regime $\alpha \to \infty$, as may be seen in the inset of Figure 2.

## 4   Discussion

We analyzed the typical learning behavior of a toy perceptron model that allows to clarify some aspects of generalization in high dimensional feature spaces. In particular, it captures an element essential to obtain stepwise learning, which is shown to stem from the compression of high order features. The components in the compressed space are more difficult to learn than those not compressed. Thus, if

the training set is not large enough, mainly the latter are learnt.

Our results allow to understand the importance of the scaling of high order features in the SVMs kernels. In fact, with SVMs one has to choose *a priori* the kernel that maps the input space to the feature space. If high order features are conveniently compressed, hierarchical learning occurs. That is, low order features are learnt first; higher order features are only learnt if the training set is large enough. In the cases where the higher order features are irrelevant, it is likely that they will not hinder the learning process. This interesting behavior allows to avoid overfitting. Computer simulations currently in progress, of SVMs generated by quadratic kernels with and without the $1/N$ scaling, show a behavior consistent with the theoretical predictions [2, 3]. These may be understood with the present toy model.

# References

[1] V. Vapnik (1995) The nature of statistical learning theory. Springer Verlag, New York.

[2] R. Dietrich, M. Opper, and H. Sompolinsky (1999) Statistical Mechanics of Support Vector Networks. Phys. Rev. Lett. 82, 2975-2978.

[3] A. Buhot and M. B. Gordon (1999) Statistical mechanics of support vector machines. *ESANN'99-European Symposium on Artificial Neural Networks* Proceedings, Michel Verleysen ed. 201-206; A. Buhot and M. B. Gordon (1998) Learning properties of support vector machines. Cond-Mat/9802179.

[4] H. Yoon and J.-H. Oh (1998) Learning of higher order perceptrons with tunable complexities J. Phys. A: Math. Gen. 31, 7771-7784.

[5] G. Györgyi and N. Tishby (1990) Statistical Theory of Learning a Rule. In Neural Networks and Spin Glasses (W. K. Theumann and R. Köberle, Worls Scientific), 3-36.

[6] R. Meir (1995) Empirical risk minimizaton. A case study. Neural Comp. 7, 144-157.

[7] C. Marangi, M. Biehl, S. A. Solla (1995) Supervised Learning from Clustered Examples Europhys. Lett. 30 (2), 117-122.
